# Learning Attractor Landscapes for Learning Motor Primitives

**Auke Jan Ijspeert**[1,3*]**, Jun Nakanishi**[2]**, and Stefan Schaal**[1,2]
[1]University of Southern California, Los Angeles, CA 90089-2520, USA
[2]ATR Human Information Science Laboratories, Kyoto 619-0288, Japan
[3]EPFL, Swiss Federal Institute of Technology, Lausanne, Switzerland
*ijspeert@usc.edu, jun@his.atr.co.jp, sschaal@usc.edu*

## Abstract

Many control problems take place in continuous state-action spaces, e.g., as in manipulator robotics, where the control objective is often defined as finding a desired trajectory that reaches a particular goal state. While reinforcement learning offers a theoretical framework to learn such control policies from scratch, its applicability to higher dimensional continuous state-action spaces remains rather limited to date. Instead of learning from scratch, in this paper we suggest to learn a desired complex control policy by transforming an existing simple canonical control policy. For this purpose, we represent canonical policies in terms of differential equations with well-defined attractor properties. By nonlinearly transforming the canonical attractor dynamics using techniques from nonparametric regression, almost arbitrary new nonlinear policies can be generated without losing the stability properties of the canonical system. We demonstrate our techniques in the context of learning a set of movement skills for a humanoid robot from demonstrations of a human teacher. Policies are acquired rapidly, and, due to the properties of well formulated differential equations, can be re-used and modified on-line under dynamic changes of the environment. The linear parameterization of nonparametric regression moreover lends itself to recognize and classify previously learned movement skills. Evaluations in simulations and on an actual 30 degree-of-freedom humanoid robot exemplify the feasibility and robustness of our approach.

## 1 Introduction

Learning control is formulated in one of the most general forms as learning a control policy $\mathbf{u} = \pi(\mathbf{x}, t, \mathbf{w})$ that maps a state $\mathbf{x}$, possibly in a time $t$ dependent way, to an action $\mathbf{u}$; the vector $\mathbf{w}$ denotes the adjustable parameters that can be used to optimize the policy. Since learning control policies (CPs) based on atomic state-action representations is rather time consuming and faces problems in higher dimensional and/or continuous state-action spaces, a current topic in learning control is to use

higher level representations to achieve faster and more robust learning [1, 2]. In this paper we suggest a novel encoding for such higher level representations based on the analogy between CPs and differential equations: both formulations suggest a change of state given the current state of the system, and both usually encode a desired goal in form of an attractor state. Thus, instead of shaping the attractor landscape of a policy tediously from scratch by traditional methods of reinforcement learning, we suggest to start out with a differential equation that already encodes a rough form of an attractor landscape and to only adapt this landscape to become more suitable to the current movement goal. If such a representation can keep the policy linear in the parameters $\mathbf{w}$, rapid learning can be accomplished, and, moreover, the parameter vector may serve to classify a particular policy.

In the following sections, we will first develop our learning approach of shaping attractor landscapes by means of statistical learning building on preliminary previous work [3, 4].[1] Second, we will present a particular form of canonical CPs suitable for manipulator robotics, and finally, we will demonstrate how our methods can be used to classify movement and equip an actual humanoid robot with a variety of movement skills through imitation learning.

## 2   Learning Attractor Landscapes

We consider a learning scenario where the goal of control is to attain a particular attractor state, either formulated as a point attractor (for discrete movements) or as a limit cycle (for rhythmic movements). For point attractors, we require that the CP will reach the goal state with a particular trajectory shape, irrespective of the initial conditions — a tennis swing toward a ball would be a typical example of such a movement. For limit cycles, the goal is given as the trajectory shape of the limit cycle and needs to be realized from any start state, as for example, in a complex drumming beat hitting multiple drums during one period. We will assume that, as the seed of learning, we obtain one or multiple example trajectories, defined by positions and velocities over time. Using these samples, an asymptotically stable CP is to be generated, prescribing a desired velocity given a particular state[2].

Various methods have been suggested to solve such control problems in the literature. As the simplest approach, one could just use one of the demonstrated trajectories and track it as a desired trajectory. While this would mimic this one particular trajectory, and scaling laws could account for different start positions [5], the resultant control policy would require time as an explicit variable and thus become highly sensitive toward unforeseen perturbations in the environment that would disrupt the normal time flow. Spline-based approaches [6] have a similar problem. Recurrent neural networks were suggested as a possible alternative that can avoid explicit time indexing — the complexity of training these networks to obtain stable attractor landscapes, however, has prevented a widespread application so far. Finally, it is also possible to prime a reinforcement learning system with sample trajectories and pursue one of the established continuous state-action learning algorithms; investigations of such an approach, however, demonstrated rather limited efficiency [7]. In the next sections, we present an alternative and surprisingly simple solution to learning the control problem above.

Table 1: Discrete and Rhythmic control policies. $\alpha_z, \beta_z, \alpha_v, \beta_v, \alpha_z, \beta_z, \mu, \sigma_i$ and $c_i$ are positive constants. $x_0$ is the start state of the discrete system in order to allow non-zero initial conditions. The design parameters of the discrete system are $\tau$, the temporal scaling factor, and $g$, the goal position. The design parameters of the rhythmic system are $y_m$, the baseline of the oscillation, $\tau$, the period divided by $2\pi$, and $r_o$, the amplitude of oscillations. The parameters $\mathbf{w}_i$ are fitted to a demonstrated trajectory using Locally Weighted Learning.

| Discrete | Rhythmic |
|---|---|
| $\tau\dot{y} = z + \frac{\sum_{i=1}^{N}\Psi_i \mathbf{w}_i^T \tilde{\mathbf{v}}}{\sum_{i=1}^{N}\Psi_i}$ <br> $\tau\dot{z} = \alpha_z(\beta_z(g-y)-z)$ <br> $\tilde{\mathbf{v}} = [v]$ | $\tau\dot{y} = z + \frac{\sum_{i=1}^{N}\Psi_i \mathbf{w}_i^T \tilde{\mathbf{v}}}{\sum_{i=1}^{N}\Psi_i}$ <br> $\tau\dot{z} = \alpha_z(\beta_z(y_m-y)-z)$ <br> $\tilde{\mathbf{v}} = [r\cos\phi, r\sin\phi]^T$ |
| $\tau\dot{v} = \alpha_v(\beta_v(g-x)-v)$ <br> $\tau\dot{x} = v$ | $\tau\dot{\phi} = 1$ <br> $\tau\dot{r} = -\mu(r-r_0)$ |
| $\Psi_i = \exp\left(-h_i(\frac{x-x_0}{g-x_0}-c_i)^2\right)$ <br> $c_i \in [0,1]$ | $\Psi_i = \exp\left(-h_i(\mathrm{mod}(\phi,2\pi)-c_i)^2\right)$ <br> $c_i \in [0,2\pi]$ |

## 2.1 Dynamical systems for Discrete Movements

Assume we have a basic control policy (CP), for instance, instantiated by the second order attractor dynamics

$$\tau\dot{z} = \alpha_z(\beta_z(g-y)-z) \qquad \tau\dot{y} = z + f \qquad (1)$$

where $g$ is a known goal state, $\alpha_z, \beta_z$ are time constants, $\tau$ is a temporal scaling factor (see below) and $y, \dot{y}$ correspond to the desired position and velocity generated by the policy as a movement plan. For appropriate parameter settings and $f = 0$, these equations form a globally stable linear dynamical system with $g$ as a unique point attractor. Could we insert a nonlinear function $f$ in Eq.1 to change the rather trivial exponential convergence of $y$ to allow more complex trajectories on the way to the goal? As such a change of Eq.1 enters the domain of nonlinear dynamics, an arbitrary complexity of the resulting equations can be expected. To the best of our knowledge, this has prevented research from employing generic learning in nonlinear dynamical systems so far. However, the introduction of an additional canonical dynamical system $(x, v)$

$$\tau\dot{v} = \alpha_v(\beta_v(g-x)-v) \qquad \tau\dot{x} = v \qquad (2)$$

and the nonlinear function $f$

$$f(x,v,g) = \frac{\sum_{i=1}^{N}\Psi_i w_i v}{\sum_{i=1}^{N}\Psi_i} \qquad \Psi_i = \exp\left(-h_i(x/g-c_i)^2\right) \qquad (3)$$

can alleviate this problem. Eq.2 is a second order dynamical system similar to Eq.1, however, it is linear and not modulated by a nonlinear function, and, thus, its monotonic global convergence to $g$ can be guaranteed with a proper choice of $\alpha_v$ and $\beta_v$. Assuming that all initial conditions of the state variables $x, v, y, z$ are initially zero, the quotient $x/g \in [0,1]$ can serve as a phase variable to anchor the Gaussian basis functions $\Psi_i$ (characterized by a center $c_i$ and bandwidth $h_i$), and $v$ can act as a "gating term" in the nonlinear function (3) such that the influence of this function vanishes at the end of the movement. Assuming boundedness of the weights $w_i$ in Eq.3, it can be shown that the combined dynamical system (Eqs.1–3) asymptotically converges to the unique point attractor $g$.

Given that $f$ is a normalized basis function representation with linear parameterization, it is obvious that this choice of a nonlinearity allows applying a variety of

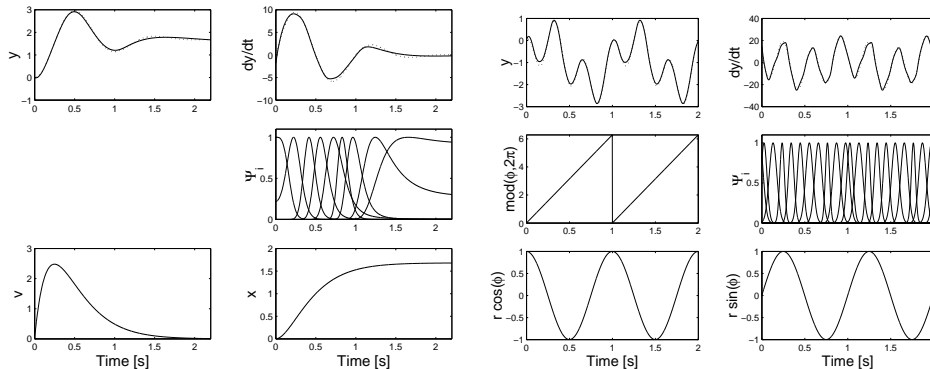

Figure 1: Examples of time evolution of the discrete CPs (left) and rhythmic CPs (right). The parameters $\mathbf{w}_i$ have been adjusted to fit $\dot{y}_{demo}(t) = 10\sin(2\pi t)\exp(-t^2)$ for the discrete CPs and $\dot{y}_{demo}(t) = 2\pi\cos(2\pi t) - 6\pi\sin(6\pi t)$ for the rhythmic CPs.

learning algorithms to find the $w_i$. For learning from a given sample trajectory, characterized by a trajectory $y_{demo}(t), \dot{y}_{demo}(t)$ and duration $T$, a supervised learning problem can be formulated with the target trajectory $f_{target} = \tau\dot{y}_{demo} - z_{demo}$ for Eq.1 (right), where $z_{demo}$ is obtained by integrating Eq.1 (left) with $y_{demo}$ instead of $y$. The corresponding goal state is $g = y_{demo}(T) - y_{demo}(t = 0)$, i.e., the sample trajectory was translated to start at $y = 0$. In order to make the nominal (i.e., assuming $f = 0$) dynamics of Eqs.1 and 2 span the duration $T$ of the sample trajectory, the temporal scaling factor $\tau$ is adjusted such that the nominal dynamics achieves 95% convergence at $t = T$. For solving the function approximation problem, we chose a nonparametric regression technique from locally weighted learning (LWL) [8] as it allows us to determine the necessary number of basis functions, their centers $c_i$, and bandwidth $h_i$ automatically — in essence, for every basis function $\Psi_i$, LWL performs a locally weighted regression of the training data to obtain an approximation of the tangent of the function to be approximated within the scope of the kernel, and a prediction for a query point is achieved by a $\Psi_i$-weighted average of the predictions all local models. Moreover, as will be explained later, the parameters $w_i$ learned by LWL are also independent of the number of basis functions, such that they can be used robustly for categorization of different learned CPs.

In summary, by anchoring a linear learning system with nonlinear basis functions in the *phase space* of a *canonical dynamical system with guaranteed attractor properties*, we are able to learn complex attractor landscapes of nonlinear differential equations without losing the asymptotic convergence to the goal state.

## 2.2   Extension to Limit Cycle Dynamics

The system above can be extended to limit cycle dynamics by replacing the canonical system $(x, v)$ with, for instance, the following rhythmic system which has a stable limit cycle in terms of polar coordinates $(\phi, r)$:

$$\tau\dot{\phi} = 1 \qquad \tau\dot{r} = -\mu(r - r_0) \qquad (4)$$

Similar to the discrete system, the rhythmic canonical system serves to provide both an amplitude signal $\tilde{\mathbf{v}} = [r\cos\phi, r\sin\phi]^T$ and phase variable $\mathrm{mod}(\phi, 2\pi)$ to the basis function $\Psi_i$ of the control policy $(z, y)$:

$$\tau\dot{z} = \alpha_z(\beta_z(y_m - y) - z) \qquad \tau\dot{y} = z + \frac{\sum_{i=1}^{N}\Psi_i\mathbf{w}_i^T\tilde{\mathbf{v}}}{\sum_{i=1}^{N}\Psi_i} \qquad (5)$$

where $y_m$ is an anchor point for the oscillatory trajectory. Table 1 summarizes the proposed discrete and rhythmic CPs, and Figure 1 shows exemplary time evolutions of the complete systems.

## 2.3 Special Properties of Control Policies based on Dynamical Systems

**Spatial and Temporal Invariance** An interesting property of both discrete and rhythmic CPs is that they are spatially and temporally invariant. Scaling of the goal $g$ for the discrete CP and of the amplitude $r_0$ for the rhythmic CP does not affect the topology of the attractor landscape. Similarly, the period (for the rhythmic system) and duration (for the discrete system) of the trajectory $y$ is directly determined by the parameter $\tau$. This means that the amplitude and durations/periods of learned patterns can be independently modified without affecting the qualitative shape of trajectory $y$. In section 3, we will exploit these properties to reuse a learned movement (such as a tennis swing, for instance) in novel conditions (e.g toward new ball positions).

**Robustness against Perturbations** When considering applications of our approach to physical systems, e.g., robots and humanoids, interactions with the environment may require an on-line modification of the policy. An obstacle can, for instance, block the trajectory of the robot, in which case large discrepancies between desired positions generated by the control policy and actual positions of the robot will occur. As outlined in [3], the dynamical system formulation allows feeding back an error term between actual and desired positions into the CPs, such that the time evolution of the policy is smoothly paused during a perturbation, i.e., the desired position $y$ is modified to remain close to the actual position $\tilde{y}$. As soon as the perturbation stops, the CP rapidly resumes performing the (time-delayed) planned trajectory. Note that other (task-specific) ways to cope with perturbations can be designed. Such on-line adaptations are one of the most interesting properties of using autonomous differential equations for CPs.

**Movement Recognition** Given the temporal and spatial invariance of our policy representation, trajectories that are topologically similar tend to be fit by similar parameters $w_i$, i.e., similar trajectories at different speeds and/or different amplitudes will result in similar $w_i$. In section 3.3, we will use this property to demonstrate the potential of using the CPs for movement recognition.

## 3 Experimental Evaluations

### 3.1 Learning of Rhythmic Control Policies by Imitation

We tested the proposed CPs in a learning by demonstration task with a humanoid robot. The robot is a 1.9-meter tall 30 DOFs hydraulic anthropomorphic robot with legs, arms, a jointed torso, and a head [9]. We recorded trajectories performed by a human subject using a joint-angle recording system, the Sarcos Sensuit (see Figure 2, top). The joint-angle trajectories are fitted by the CPs, with one CP per degree of freedom (DOF). The CPs are then used to replay the movement in the humanoid robot, using an inverse dynamics controller to track the desired trajectories generated by the CPs. The actual positions $\tilde{y}$ of each DOF are fed back into the CPs in order to take perturbations into account.

Using the joint-angle recording system, we recorded a set of rhythmic movements such as tracing a figure 8 in the air, or a drumming sequence on a bongo (i.e. without drumming sticks). Six DOFs for both arms were recorded (three at the shoulder, one at the elbow, and two at the wrist). An exemplary movement and its replication by the robot is demonstrated in Figure 2 (top). Figure 2 (left) shows the joint trajectories over one period of an exemplary drumming beat. Demonstrated and learned trajectories are superposed. For the learning, the base frequency was extracted by hand such as to provide the parameter $\tau$ to the rhythmic CP.

Once a rhythmic movement has been learned by the CP, it can be modulated in several ways. Manipulating $r_0$ and $\tau$ for all DOFs amounts to simultaneously

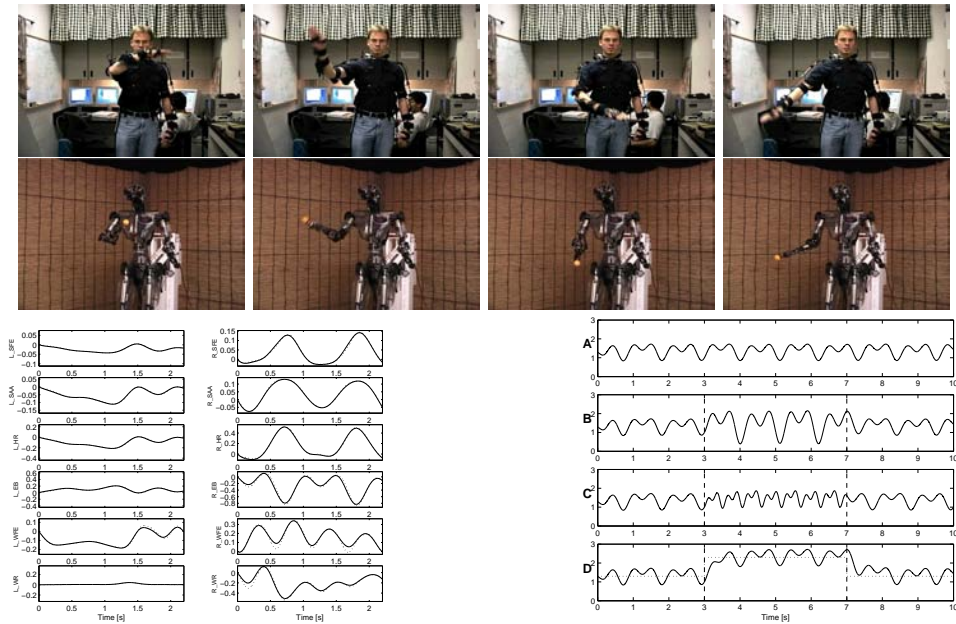

Figure 2: **Top:** Humanoid robot learning a figure-8 movement from a human demonstration. **Left:** Recorded drumming movement performed with both arms (6 DOFs per arm). The dotted lines and continuous lines correspond to one period of the demonstrated and learned trajectories, respectively. **Right:** Modification of the learned rhythmic pattern (flexion/extension of the right elbow, R_EB). **A:** trajectory learned by the rhythmic CP, **B:** temporary modification with $\tilde{r}_0 = 2r_0$, **C:** $\tilde{\tau} = \tau/2$, **D:** $\tilde{y_m} = y_m + 1$ (dotted line), where $\tilde{r}_0$, $\tilde{\tau}$, and $\tilde{y_m}$ correspond to modified parameters between t=3s and t=7s. Movies of the human subject and the humanoid robot can be found at http://lslwww.epfl.ch/~ijspeert/humanoid.html.

modulate the amplitude and period of all DOFs, while keeping the same phase relation between DOFs. This might be particularly useful for a drumming task in order to replay the same beat pattern at different speeds and/or amplitudes. Alternatively, the $r_0$ and $\tau$ parameters can be modulated independently for the DOFs each arm, in order to be able to change the beat pattern (doubling the frequency of one arm, for instance). Figure 2 (right) illustrates different modulations which can be generated by the rhythmic CPs. For reasons of clarity, only one DOF is showed. The rhythmic CP can smoothly modulate the amplitude, frequency, and baseline of the oscillations.

## 3.2 Learning of Discrete Control Policies by Imitation

In this experiment, the task for the robot was to learn tennis forehand and backhand swings demonstrated by a human wearing the joint-angle recording system. Once a particular swing has been learned, the robot is able to repeat the swing motion to different cartesian targets, by providing new goal positions $g$ to the CPs for the different DOFs. Using a system of two-cameras, the position of the ball is given to an inverse kinematic algorithm which computes these new goals in joint space. When the new ball positions are not too distant from the original cartesian target, the modified trajectories reach the ball with swing motions very similar to those used for the demonstration.

## 3.3 Movement Recognition using the Discrete Control Policies

Our learning algorithm, Locally Weighted Learning [8], automatically sets the number of the kernel functions and their centers $c_i$ and widths $h_i$ depending on the complexity of the function to be approximated, with more kernel functions for highly

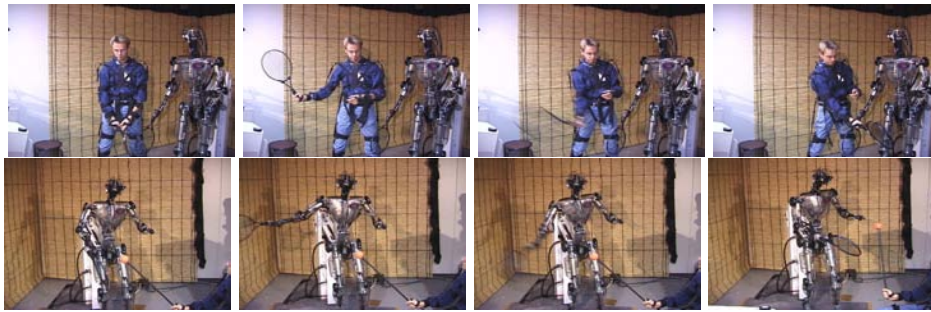

Figure 3: Humanoid robot learning a forehand swing from a human demonstration.

nonlinear details of the movement. An interesting aspect of locally weighted regression is that the regression parameters $\mathbf{w}_i$ of each kernel $i$ do *not* depend on the other kernels, since regression is based on a separate cost function for each kernel. This means that kernel functions can be added or removed without affecting the parameters $\mathbf{w}_i$ of the other kernels.

We here use this feature to perform movement recognition within a large variety of trajectories, based on a small subset of kernels at fixed locations $c_i$ in phase space. These fixed kernels are common for fitting all the trajectories, in addition to the kernels automatically added by the LWL algorithm. The stability of their parameters $\mathbf{w}_i$ w.r.t. other kernels generated by LWL makes them well-suited for comparing qualitative trajectory shapes.

To illustrate the possibility of using the CPs for movement recognition (i.e., recognition of spatiotemporal patterns, not just spatial patterns as in traditional character recognition), we carried out a simple task of fitting trajectories performed by a human user when drawing two-dimensional single-stroke patterns. The 26 letters of the Graffiti alphabet used in hand-held computers were chosen. These characters are drawn in a single stroke, and are fed as a two-dimensional trajectory $(x(t), y(t))$ to be fitted by our system. Five examples of each character were presented (see Figure 4 for four examples).

Fixed sets of five kernels per DOF were set aside for movement recognition. The correlation $\frac{\mathbf{w}_a^T \mathbf{w}_b}{|\mathbf{w}_a||\mathbf{w}_b|}$ between their parameter vectors $\mathbf{w}_a$ and $\mathbf{w}_b$ of character $a$ and $b$ can be used to classify movements with similar velocity profiles (Figure 4, right). For instance, for the 5 instances of the N, I, P, S, characters, the correlation is systematically higher with the four other examples of the same character. These similarities in weight space can therefore serve as basis for recognizing demonstrated movements by fitting them and comparing the fitted parameters $w_i$ with those of previously learned policies in memory. In this example, a simple one-nearest-neighbor classifier in weight space would serve the purpose. Using such a classifier within the whole alphabet (5 instances of each letter), we obtained a 84% recognition rate (i.e. 110 out of the 130 instances had a highest correlation with an instance of the same letter). Further studies are required to evaluate the quality of recognition in larger training and test sets — what we wanted to demonstrate is the ability for recognition without any specific system tuning or sophisticated classification algorithm.

## 4  Conclusion

Based on the analogy between autonomous differential equations and control policies, we presented a novel approach to learn control policies of basic movement skills by shaping the attractor landscape of nonlinear differential equations with statistical learning techniques. To the best of our knowledge, the presented approach is the first realization of a generic learning system for nonlinear dynamical systems that

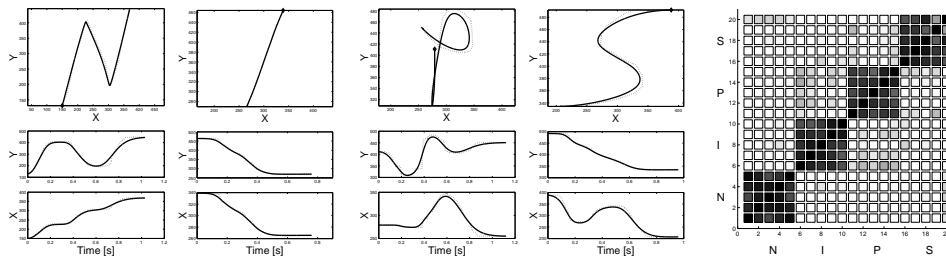

Figure 4: **Left:** Examples of two-dimensional trajectories fitted by the CPs. The demonstrated and fitted trajectories are shown with dotted and continuous lines, respectively. **Right:** Correlation between the weight vectors of the 20 characters (5 of each letter) fitted by the system. The gray scale is proportional to the correlation, with black corresponding to a correlation of +1 (max. correlation) and white to a correlation of 0 or smaller.

can guarantee basic stability and convergence properties of the learned nonlinear systems. We demonstrated the applicability of the suggested techniques by learning various movement skills for a complex humanoid robot by imitation learning, and illustrated the usefulness of the learned parameterization for recognition and classification of movement skills. Future work will consider (1) learning of multidimensional control policies without assuming independence between the individual dimensions, and (2) the suitability of the linear parameterization of the control policies for reinforcement learning.

**Acknowledgments**
This work was made possible by support from the US National Science Foundation (Awards 9710312 and 0082995), the ERATO Kawato Dynamic Brain Project funded by the Japan Science and Technology Corporation, the ATR Human Information Science Laboratories, and Communications Research Laboratory (CRL).

## Footnotes

*http://lslwww.epfl.ch/~ijspeert/

[1]Portions of the work presented in this paper have been published in [3, 4]. We here extend these preliminary studies with an improvement and simplification of the rhythmic system, an integrated view of the interpretation of both the discrete and rhythmic CPs, the fitting of a complete alphabet of Grafitti characters, and an implementation of automatic allocation of centers of kernel functions for locally weighted learning.

[2]Note that we restrict our approach to purely kinematic CPs, assuming that the movement system is equipped with an appropriate feedback and feedforward controller that can accurately track the kinematic plans generated by our policies.

# References

[1] R. Sutton and A.G. Barto. *Reinforcement learning: an introduction*. MIT Press, 1998.

[2] F.A. Mussa-Ivaldi. Nonlinear force fields: a distributed system of control primitives for representing and learning movements. In *IEEE International Symposium on Computational Intelligence in Robotics and Automation*, pages 84–90. IEEE, Computer Society, Los Alamitos, 1997.

[3] A.J. Ijspeert, J. Nakanishi, and S. Schaal. Movement imitation with nonlinear dynamical systems in humanoid robots. In *IEEE International Conference on Robotics and Automation (ICRA2002)*, pages 1398–1403. 2002.

[4] A.J. Ijspeert, J. Nakanishi, and S. Schaal. Learning rhythmic movements by demonstration using nonlinear oscillators. In *Proceedings of the IEEE/RSJ Int. Conference on Intelligent Robots and Systems (IROS2002)*, pages 958–963. 2002.

[5] S. Kawamura and N. Fukao. Interpolation for input torque patterns obtained through learning control. In *Proceedings of The Third International Conference on Automation, Robotics and Computer Vision (ICARCV'94)*. 1994.

[6] H. Miyamoto, S. Schaal, F. Gandolfo, Y. Koike, R. Osu, E. Nakano, Y. Wada, and M. Kawato. A kendama learning robot based on bi-directional theory. *Neural Networks*, 9:1281–1302, 1996.

[7] S. Schaal. Learning from demonstration. In M. C. Mozer, M. Jordan, and T. Petsche, editors, *Advances in Neural Information Processing Systems 9*, pages 1040–1046. Cambridge, MA, MIT Press, 1997.

[8] S. Schaal and C.G. Atkeson. Constructive incremental learning from only local information. *Neural Computation*, 10(8):2047–2084, 1998.

[9] C. G. Atkeson, J. Hale, M. Kawato, S. Kotosaka, F. Pollick, M. Riley, S. Schaal, S. Shibata, G. Tevatia, A. Ude, and S. Vijayakumar. Using humanoid robots to study human behavior. *IEEE Intelligent Systems*, 15:46–56, 2000.
